# A Computational Decision Theory for Interactive Assistants

**Alan Fern**
School of EECS
Oregon State University
Corvallis, OR 97331
afern@eecs.oregonstate.edu

**Prasad Tadepalli**
School of EECS
Oregon State University
Corvallis, OR 97331
tadepall@eecs.oregonstate.edu

## Abstract

We study several classes of interactive assistants from the points of view of decision theory and computational complexity. We first introduce a class of POMDPs called hidden-goal MDPs (HGMDPs), which formalize the problem of interactively assisting an agent whose goal is hidden and whose actions are observable. In spite of its restricted nature, we show that optimal action selection in finite horizon HGMDPs is PSPACE-complete even in domains with deterministic dynamics. We then introduce a more restricted model called helper action MDPs (HAMDPs), where the assistant's action is accepted by the agent when it is helpful, and can be easily ignored by the agent otherwise. We show classes of HAMDPs that are complete for PSPACE and NP along with a polynomial time class. Furthermore, we show that for general HAMDPs a simple myopic policy achieves a regret, compared to an omniscient assistant, that is bounded by the entropy of the initial goal distribution. A variation of this policy is shown to achieve worst-case regret that is logarithmic in the number of goals for any goal distribution.

## 1 Introduction

Integrating AI with Human Computer Interaction has received significant attention in recent years [8, 11, 13, 3, 2]. In most applications, e.g. travel scheduling, information retrieval, or computer desktop navigation, the relevant state of the computer is fully observable, but the goal of the user is not, which poses a difficult problem to the computer assistant. The assistant needs to correctly reason about the relative merits of taking different actions in the presence of significant uncertainty about the goals of the human agent. It might consider taking actions that directly reveal the goal of the agent, e.g. by asking questions to the user. However, direct communication is often difficult due to the language mismatch between the human and the computer. Another strategy is to take actions that help achieve the most likely goals. Yet another strategy is to take actions that help with a large number of possible goals. In this paper, we formulate and study several classes of interactive assistant problems from the points of view of decision theory and computational complexity. Building on the framework of decision-theoretic assistance (DTA) [5], we analyze the inherent computational complexity of optimal assistance in a variety of settings and the sources of that complexity. Positively, we analyze a simple myopic heuristic and show that it performs nearly optimally in a reasonably pervasive assistance problem, thus explaining some of the positive empirical results of [5].

We formulate the problem of optimal assistance as solving a hidden-goal MDP (HGMDP), which is a special case of a POMDP [6]. In a HGMDP, a (human) agent and a (computer) assistant take actions in turns. The agent's goal is the only unobservable part of the state of the system and does not change throughout the episode. The objective for the assistant is to find a history-dependent policy that maximizes the expected reward of the agent given the agent's goal-based policy and its goal distribution. Despite the restricted nature of HGMDPs, the complexity of determining if an HGMDP has a finite-horizon policy of a given value is PSPACE-complete even in deterministic

environments. This motivates a more restricted model called Helper Action MDP (HAMDP), where the assistant executes a helper action at each step. The agent is obliged to accept the helper action if it is helpful for its goal and receives a reward bonus (or cost reduction) for doing so. Otherwise, the agent can continue with its own preferred action without any reward or penalty to the assistant. We show classes of this problem that are complete for PSPACE and NP. We also show that for the class of HAMDPs with deterministic agents there are polynomial time algorithms for minimizing the expected and worst-case regret relative to an omniscient assistant. Further, we show that the optimal worst case regret can be characterized by a graph-theoretic property called the tree rank of the corresponding all-goals policy tree and can be computed in linear time.

The main positive result of the paper is to give a simple myopic policy for general stochastic HAMDPs that has a regret which is upper bounded by the entropy of the goal distribution. Furthermore we give a variant of this policy that is able to achieve worst-case and expected regret that is logarithmic in the number of goals without any prior knowledge of the goal distribution.

To the best of our knowledge, this is the first formal study of the computational hardness of the problem of decision-theoretically optimal assistance and the performance of myopic heuristics. While the current HAMDP results are confined to unobtrusively assisting a competent agent, they provide a strong foundation for analyzing more complex classes of assistant problems, possibly including direct communication, coordination, partial observability, and irrationality of users.

## 2  Hidden Goal MDPs

Throughout the paper we will refer to the entity that we are attempting to assist as the *agent* and the assisting entity as the *assistant*. Our objective is to select actions for the assistant in order to help the agent maximize its reward. The key complication is that the agent's goal is not directly observable to the assistant, so reasoning about the likelihood of possible goals and how to help maximize reward given those goals is required. In order to support this type of reasoning we will model the agent-assistant process via *hidden goal MDPs (HGMDPs)*.

**General Model.** An HGMDP describes the dynamics and reward structure of the environment via a first-order Markov model, where it is assumed that the state is fully observable to both the agent and assistant. In addition, an HGMDP describes the possible goals of the agent and the behavior of the agent when pursuing those goals. More formally, an HGMDP is a tuple $\langle S, G, A, A', T, R, \pi, I_S, I_G \rangle$ where $S$ is a set of states, $G$ is a finite set of possible agent goals, $A$ is the set of agent actions, $A'$ is the set of assistant actions, $T$ is the transition function such that $T(s, g, a, s')$ is the probability of a transition to state $s'$ from $s$ after taking action $a \in A \cup A'$ when the agent goal is $g$, $R$ is the reward function which maps $S \times G \times (A \cup A')$ to real valued rewards, $\pi$ is the agent's policy that maps $S \times G$ to distributions over $A$ and need not be optimal in any sense, and $I_S$ ($I_G$) is an initial state (goal) distribution. The dependence of the reward and policy on the goal allows the model to capture the agent's desires and behavior under each goal. The dependence of $T$ on the goal is less intuitive and in many cases there will be no dependence when $T$ is used only to model the dynamics of the environment. However, we allow goal dependence of $T$ for generality of modeling. For example, it can be convenient to model basic communication actions of the agent as changing aspects of the state, and the result of such actions will often be goal dependent.

We consider a finite-horizon episodic problem setting where the agent begins each episode in a state drawn from $I_S$ with a goal drawn from $I_G$. The goal, for example, might correspond to a physical location, a dish that the agent wants to cook, or a destination folder on a computer desktop. The process then alternates between the agent and assistant executing actions (including noops) in the environment until the horizon is reached. The agent is assumed to select actions according to $\pi$. In many domains, a terminal goal state will be reached within the horizon, though in general, goals can have arbitrary impact on the reward function. The reward for the episode is equal to the sum of the rewards of the actions executed by the agent and assistant during the episode. The objective of the assistant is to reason about the HGMDP and observed state-action history in order to select actions that maximize the expected (or worst-case) total reward of an episode.

An example HGMDP from previous work [5] is the doorman domain, where an agent navigates a grid world in order to arrive at certain goal locations. To move from one location to another the agent must open a door and then walk through the door. The assistant can reduce the effort for the agent by opening the relevant doors for the agent. Another example from [1] involves a computer

desktop where the agent wishes to navigate to certain folders using a mouse. The assistant can select actions that offer the agent a small number of shortcuts through the folder structure.

Given knowledge of the agent's goal $g$ in an HGMDP, the assistant's problem reduces to solving an MDP over assistant actions. The MDP transition function captures both the state change due to the assistant action and also the ensuing state change due to the agent action selected according to the policy $\pi$ given $g$. Likewise the reward function on a transition captures the reward due to the assistant action and the ensuing agent action conditioned on $g$. The optimal policy for this MDP corresponds to an optimal assistant policy for $g$. However, since the real assistant will often have uncertainty about the agent's goal, it is unlikely that this optimal performance will be achieved.

**Computational Complexity.** We can view an HGMDP as a collection of $|G|$ MDPs that share the same state space, where the assistant is placed in one of the MDPs at the beginning of each episode, but cannot observe which one. Each MDP is the result of fixing the goal component of the HG-MDP definition to one of the goals. This collection can be easily modeled as a restricted type of partially observable MDP (POMDP) with a state space $S \times G$. The $S$ component is completely observable, while the $G$ component is unobservable but only changes at the beginning of each episode (according to $I_G$) and remains constant throughout an episode. Furthermore, each POMDP transition provides observations of the agent action, which gives direct evidence about the unchanging $G$ component. From this perspective HGMDPs appear to be a significant restriction over general POMDPs. However, our first result shows that despite this restriction the worst-case complexity is not reduced even for deterministic dynamics.

Given an HGMDP $M$, a horizon $m = O(|M|)$ where $|M|$ is the size of the encoding of $M$, and a reward target $r^*$, the *short-term reward maximization problem* asks whether there exists a history-dependent assistant policy that achieves an expected finite horizon reward of at least $r^*$. For general POMDPs this problem is PSPACE-complete [12, 10], and for POMDPs with deterministic dynamics, it is NP-complete [9]. However, we have the following result.

**Theorem 1.** *Short-term reward maximization for HGMDPs with deterministic dynamics is PSPACE-complete.*

The proof is in the appendix. This result shows that any POMDP can be encoded as an HGMDP with deterministic dynamics, where the stochastic dynamics of the POMDP are captured via the stochastic agent policy in the HGMDP. However, the HGMDPs resulting from the PSPACE-hardness reduction are quite pathological compared to those that are likely to arise in practice. Most importantly, the agent's actions provide practically no information about the agent's goal until the end of an episode, when it is too late to exploit this knowledge. This suggests that we search for restricted classes of HGMDPs that will allow for efficient solutions with performance guarantees.

## 3 Helper Action MDPs

The motivation for HAMDPs is to place restrictions on the agent and assistant that avoid the following three complexities that arise in general HGMDPs: 1) the agent can behave arbitrarily poorly if left unassisted and as such the agent actions may not provide significant evidence about the goal; 2) the agent is free to effectively "ignore" the assistant's help and not exploit the results of assistive action, even when doing so would be beneficial; and 3) the assistant actions have the possibility of negatively impacting the agent compared to not having an assistant. HAMDPs will address the first issue by assuming that the agent is competent at (approximately) maximizing reward without the assistant. The last two issues will be addressed by assuming that the agent will always "detect and exploit" helpful actions and that the assistant actions do not hurt the agent.

Informally, the HAMDP provides the assistant with a *helper action* for each of the agent's actions. Whenever a helper action $h$ is executed directly before the corresponding agent action $a$, the agent receives a bonus reward of 1. However, the agent will only accept the helper action $h$ (by taking $a$) and hence receive the bonus, if $a$ is an action that the agent considers to be good for achieving the goal without the assistant. Thus, the primary objective of the assistant in an HAMDP is to maximize the number of helper actions that get accepted by the agent. While simple, this model captures much of the essence of assistance domains where assistant actions cause minimal harm and the agent is able to detect and accept good assistance when it arises.

An HAMDP is an HGMDP $\langle S, G, A, A', T, R, \pi, I_S, I_G \rangle$ with the following constraints:

- The agent and the assistant actions sets are $A = \{a_1, \ldots, a_n\}$ and $A' = \{h_1, \ldots, h_n\}$, so that for each $a_i$ there is a corresponding *helper action* $h_i$.

- The state space is $S = W \cup (W \times A')$, where $W$ is a set of *world states*. States in $W \times A'$ encode the current world state and the previous assistant action.

- The reward function $R$ is 0 for all assistant actions. For agent actions the reward is zero unless the agent selects the action $a_i$ in state $(s, h_i)$ which gives a reward of 1. That is, the agent receives a bonus of 1 whenever its action corresponds to the preceding helper action.

- The assistant always acts from states in $W$, and $T$ is such that taking $h_i$ in $s$ deterministically transitions to $(s, h_i)$.

- The agent always acts from states in $S \times A'$, resulting in states in $S$ according to a transition function that does not depend on $h_i$, i.e. $T((s, h_i), g, a_i, s') = T'(s, g, a_i, s')$ for some transition function $T'$.

- Finally, for the agent policy, let $\Pi(s, g)$ be a function that returns a set of actions and $P(s, g)$ be a distribution over those actions. We will view $\Pi(s, g)$ as the set of actions that the agent considers acceptable (or equally good) in state $s$ for goal $g$. The agent policy $\pi$ always selects $a_i$ after its helper action $h_i$ whenever $a_i$ is acceptable. That is, $\pi((s, h_i), g) = a_i$ whenever $a_i \in \Pi(s, g)$. Otherwise the agent draws an action according to $P(s, g)$.

In a HAMDP, the primary impact of an assistant action is to influence the reward of the following agent action. The only rewards in HAMDPS are the bonuses received whenever the agent accepts a helper action. Any additional environmental reward is assumed to be already captured by the agent policy via $\Pi(s, g)$ that contains actions that approximately optimize this reward.

The HAMDP model can be adapted to both the doorman domain in [5] and the folder prediction domain from [1]. In the doorman domain, the helper actions correspond to opening doors for the agent, which reduce the cost of navigating from one room to another. Importantly opening an incorrect door has a fixed reward loss compared to an optimal assistant, which is a key property of HAMDPs. In the folder prediction domain, the system proposes multiple folders to save a file, potentially saving the user a few clicks every time the proposal is accepted.

Despite the apparent simplification of HAMDPs over HGMDPs, somewhat surprisingly the worst case computational complexity is not reduced.

**Theorem 2.** *Short-term reward maximization for HAMDPs is PSPACE-complete.*

The proof is in the appendix. Unlike the case of HGMDPs, we will see that the stochastic dynamics are essential for PSPACE-hardness. Despite this negative result, the following sections show the utility of the HAMDP restriction by giving performance guarantees for simple policies and improved complexity results in special cases. So far, there are no analogous results for HGMDPs.

## 4 Regret Analysis for HAMDPs

Given an assistant policy $\pi'$, the regret of a particular episode is the extra reward that an omniscient assistant with knowledge of the goal would achieve over $\pi'$. For HAMDPs the omniscient assistant can always achieve a reward equal to the finite horizon $m$, because it can always select a helper action that will be accepted by the agent. Thus, the regret of an execution of $\pi'$ in a HAMDP is equal to the number of helper actions that are not accepted by the agent, which we will call *mispredictions*. From above we know that optimizing regret is PSPACE-hard and thus here we focus on bounding the expected and worst-case regret of the assistant. We now show that a simple myopic policy is able to achieve regret bounds that are logarithmic in the number of goals.

**Myopic Policy.** Intuitively, our myopic assistant policy $\hat{\pi}$ will select an action that has the highest probability of being accepted with respect to a "coarsened" version of the posterior distribution over goals. The myopic policy in state $s$ given history $H$ is based on the *consistent goal set* $C(H)$, which is the set of goals that have non-zero probability with respect to history $H$. It is straightforward to maintain $C(H)$ after each observation. The myopic policy is defined as:

$$\hat{\pi}(s, H) = \arg \max_a I_G(C(H) \cap G(s, a))$$

where $G(s, a) = \{g \mid a \in \Pi(s, g)\}$ is the set of goals for which the agent considers $a$ to be an acceptable action in state $s$. The expression $I_G(C(H) \cap G(s, a))$ can be viewed as the probability

mass of $G(s, a)$ under a coarsened goal posterior which assigns goals outside of $C(H)$ probability zero and otherwise weighs them proportional to the prior.

**Theorem 3.** *For any HAMDP the expected regret of the myopic policy is bounded above by the entropy of the goal distribution $\mathcal{H}(I_G)$.*

*Proof.* The main idea of the proof is to show that after each misprediction of the myopic policy (i.e. the selected helper action is not accepted by the agent) the uncertainty about the goal is reduced by a constant factor, which will allow us to bound the total number of mispredictions on any trajectory.

Consider a misprediction step where the myopic policy selects helper action $h_i$ in state $s$ given history $H$, but the agent does not accept the action and instead selects $a^* \neq a_i$. By the definition of the myopic policy we know that $I_G(C(H) \cap G(s, a_i)) \geq I_G(C(H) \cap G(s, a^*))$, since otherwise the assistant would not have chosen $h_i$. From this fact we now argue that $I_G(C(H')) \leq I_G(C(H))/2$ where $H'$ is the history after the misprediction. That is, the probability mass under $I_G$ of the consistent goal set after the misprediction is less than half that of the consistent goal set before the misprediction. To show this we will consider two cases: 1) $I_G(C(H) \cap G(s, a_i)) < I_G(C(H))/2$, and 2) $I_G(C(H) \cap G(s, a_i)) \geq I_G(C(H))/2$. In the first case, we immediately get that $I_G(C(H) \cap G(s, a^*)) < I_G(C(H))/2$. Combining this with the fact that $C(H') \subseteq C(H) \cap G(s, a^*)$ we get the desired result that $I_G(C(H')) \leq I_G(C(H))/2$. In the second case, note that

$$C(H') \subseteq C(H) \cap (G(s, a^*) - G(s, a_i)) \subseteq C(H) - (C(H) \cap G(s, a_i))$$

Combining this with our assumption for the second case implies that $I_G(C(H')) \leq I_G(C(H))/2$. This implies that for any episode, after $n$ mispredictions resulting in a history $H_n$, $I_G(C(H_n)) \leq 2^{-n}$. Now consider an arbitrary episode where the true goal is $g$. We know that $I_G(g)$ is a lower bound on $I_G(C(H_n))$, which implies that $I_G(g) \leq 2^{-n}$ or equivalently that $n \leq -\log(I_G(g))$. Thus for any episode with goal $g$ the maximum number of mistakes is bounded by $-\log(I_G(g))$. Using this fact we get that the expected number of mispredictions during an episode with respect to $I_G$ is bounded above by $-\sum_g I_G(g) \log(I_G(g)) = \mathcal{H}(I_G)$, which completes the proof. $\qquad\square$

Since $\mathcal{H}(I_G) \leq \log(|G|)$, this result implies that for HAMDPs the expected regret of the myopic policy is no more than logarithmic in the number of goals. Furthermore, as the uncertainty about the goal decreases (decreasing $\mathcal{H}(I_G)$) the regret bound improves until we get a regret of 0 when $I_G$ puts all mass on a single goal. This logarithmic bound is asymptotically tight in the worst case.

**Theorem 4.** *There exists a HAMDP such that for any assistant policy the expected regret is at least* $\log(|G|)/2$.

*Proof.* Consider a deterministic HAMDP such that the environment is structured as a binary tree of depth $\log(|G|)$, where each leaf corresponds to one of the $|G|$ goals. By considering a uniform goal distribution it is easy to verify that at any node in the tree there is an equal chance that the true goal is in the left or right sub-tree during any episode. Thus, any policy will have a 0.5 chance of committing a misprediction at each step of an episode. Since each episode is of length $\log(|G|)$, the expected regret of an episode for any policy is $\log(|G|)/2$. $\qquad\square$

Resolving the gap between the myopic policy bound and this regret lower bound is an open problem.

**Approximate Goal Distributions.** Suppose that the assistant uses an approximate goal distribution $I'_G$ instead of the true underlying goal distribution $I_G$ when computing the myopic policy. That is, the assistant selects actions that maximize $I'_G(C(H) \cap G(s, a))$, which we will refer to as the myopic policy relative to $I'_G$. The extra regret for using $I'_G$ instead of $I_G$ can be bounded in terms of the KL-divergence between these distributions $KL(I_G \parallel I'_G)$, which is zero when $I'_G$ equals $I_G$.

**Theorem 5.** *For any HAMDP with goal distribution $I_G$, the expected regret of the myopic policy with respect to distribution $I'_G$ is bounded above by $\mathcal{H}(I_G) + KL(I_G \parallel I'_G)$.*

The proof is in the appendix. Deriving similar results for other approximations is an open problem.

A consequence of Theorem 5 is that the myopic policy with respect to the uniform goal distribution has expected regret bounded by $\log(|G|)$ for any HAMDP, showing that logarithmic regret can be achieved without knowledge of $I_G$. This can be strengthened to hold for worst case regret.

**Theorem 6.** *For any HAMDP, the worst case and hence expected regret of the myopic policy with respect to the uniform goal distribution is bounded above by* $\log(|G|)$.

*Proof.* The proof of Theorem 5 shows that the number of mispredictions on any episode is bounded above by $-\log(I'_G)$. In our case $I'_G = 1/|G|$ which shows a worst case regret bound of $\log(|G|)$, which also bounds the expected regret of the uniform myopic policy. □

## 5   Deterministic and Bounded Choice Policies

We now consider several special cases of HAMDPs. First, we restrict the agent's policy to be deterministic for each goal, i.e. $\Pi(s, g)$ has at most a single action for each state-goal pair $(s, g)$.

**Theorem 7.** *The myopic policy achieves the optimal expected reward for HAMDPs with deterministic agent policies.*

The proof is given in the appendix. We now consider the case where both the agent policy and the environment are deterministic, and attempt to minimize the worst possible regret compared to an omniscient assistant who knows the agent's goal. As it happens, this "minimax policy" can be captured by a graph-theoretic notion of tree rank that generalizes the rank of decision trees [4].

**Definition 1.** *The rank of a rooted tree is the rank of its root node. If a node is a leaf node then rank(node) = 0, else if a node has at least two distinct children $c_1$ and $c_2$ with equal highest ranks among all children, then rank(node) = 1+ rank($c_1$). Otherwise rank(node) = rank of the highest ranked child.*

The optimal trajectory tree (OTT) of a HAMDP in deterministic environments is a tree where the nodes represent the states of the HAMDP reached by the prefixes of optimal action sequences for different goals starting from the initial state.[1] Each node in the tree represents a state and a set of goals for which it is on the optimal path from the initial state.

Since the agent policy and the environment are both deterministic, there is at most one trajectory per goal in the tree. Hence the size of the optimal trajectory tree is bounded by the number of goals times the maximum length of any trajectory, which is at most the size of the state space in deterministic domains. The following Lemma follows by induction on the depth of the optimal trajectory tree.

**Lemma 1.** *The minimum worst-case regret of any policy for an HAMDP for deterministic environments and deterministic agent policies is equal to the tree rank of its optimal trajectory tree.*

**Theorem 8.** *If the agent policy is deterministic, the problem of minimizing the maximum regret in HAMDPs in deterministic environments is in P.*

*Proof.* We first construct the optimal trajectory tree. We then compute its rank and the optimal minimax policy using the recursive definition of tree rank in linear time. □

The assumption of deterministic agent policy may be too restrictive in many domains. We now consider HAMDPs in which the agent policies have a constant bound on the number of possible actions in $\Pi(s, g)$ for each state-goal pair. We call them bounded choice HAMDPs.

**Definition 2.** *The branching factor of a HAMDP is the largest number of possible actions in $\Pi(s, g)$ by the agent in any state for any goal and any assistant's action.*

The doorman domain of [5] has a branching factor of 2 since there are at most two optimal actions to reach any goal from any state.

**Theorem 9.** *Minimizing the worst-case regret in finite horizon bounded choice HAMDPS of a constant branching factor $k \geq 2$ in deterministic environments is NP-complete.*

The proof is in the appendix. We can also show that minimizing the expected regret for a bounded $k$ is NP-hard. We conjecture that this problem is also in NP, but this question remains open.

# 6 Conclusions and Future Work

In this paper, we formulated the problem of optimal assistance and analyzed its complexity in multiple settings. We showed that the general problem of HGMDP is PSPACE-complete due to the lack of constraints on the user, who can behave stochastically or even adversarially with respect to the assistant, which makes the assistant's task very difficult. By suitably constraining the user's actions through HAMDPs, we are able to reduce the complexity to NP-complete, but only in deterministic environments with bounded choice agents. More encouragingly, we are able to show that HAMDPs are amenable to a simple myopic heuristic which has a regret bounded by the entropy of the goal distribution when compared to the omniscient assistant. This is a satisfying result since optimal communication of the goal requires as much information to pass from the agent to the assistant. Importantly, this result applies to stochastic as well as deterministic environments and with no bound on the number of agent's action choices.

Although HAMDPs are somewhat restricted compared to possible assistantship scenarios one could imagine, they in fact fit naturally to many domains where the user is on-line, knows which helper actions are acceptable, and accepts help when it is appropriate to the goal. Indeed, in many domains, it is reasonable to constrain the assistant so that the agent has the final say on approving the actions proposed by the assistant. These scenarios range from the ubiquitous auto-complete functions and Microsoft's infamous Paperclip to more sophisticated adaptive programs such as SmartEdit [7] and TaskTracer [3] that learn assistant policies from users' long-term behaviors. By analyzing the complexity of these tasks in a more general framework than what is usually done, we shed light on some of the sources of complexity such as the stochasticity of the environment and the agent's policy. Many open problems remain including generalization of these and other results to more general assistant frameworks, including partially observable and adversarial settings, learning assistants, and multi-agent assistance.

# 7 Appendix

*Proof of Theorem 1.* Membership in PSPACE follows from the fact that any HGMDP can be polynomially encoded as a POMDP for which policy existence is in PSPACE. To show PSPACE-hardness, we reduce the QSAT problem to the problem of the existence of a history-dependent assistant policy of expected reward $\geq r$.

Let $\phi$ be a quantified Boolean formula $\forall x_1 \exists x_2 \forall x_3 \ldots \exists x_n \; \{C_1(x_1, \ldots, x_n) \land \ldots \land C_m(x_1, \ldots, x_n)\}$, where each $C_i$ is a disjunctive clause. For us, each goal $g_i$ is a quantified clause, $\forall x_1 \exists x_2 \forall x_3 \ldots \exists x_n \; \{C_i(x_1, \ldots, x_n)\}$. The agent chooses a goal uniformly randomly from the set of goals formed from $\phi$ and hides it from the assistant. The states consist of pairs of the form $(v, i)$, where $v \in \{0, 1\}$ is the current value of the goal clause, and $i$ is the next variable to set. The actions of the assistant are to set the existentially quantified variables. The agent simulates setting the universally quantified variables by choosing actions from the set $\{0, 1\}$ with equal probability. The episode terminates when all the variables are set, and the assistant gets a reward of 1 if the value of the clause is 1 at the end and a reward of 0 otherwise.

Note that the assistant does not get any useful feedback from the agent until it is too late and it either makes a mistake or solves the goal. The best the assistant can do is to find an optimal history-dependent policy that maximizes the expected reward over the goals in $\Phi$. If $\Phi$ is satisfiable, then there is an assistant policy that leads to a reward of 1 over all goals and all agent actions, and hence has an expected value of 1 over the goal distribution. If not, then at least one of the goals will not be satisfied for some setting of the universal quantifiers, leading to an expected value $< 1$. □

*Proof of Theorem 2.* Membership in PSPACE follows easily since HAMDP is a specialization of HGMDP. The proof of PSPACE-hardness is identical to that of 1 except that here, instead of the agent's actions, the stochastic environment models the universal quantifiers. The agent accepts all actions until the last one and sets the variable as suggested by the assistant. After each of the assistant's actions, the environment chooses a value for the universally quantified variable with equal probability. The last action is accepted by the agent if the goal clause evaluates to 1, otherwise not. There is a history-dependent policy whose expected reward $\geq$ the number of existential variables if and only if the quantified Boolean formula is satisfiable. □

*Proof of Theorem 5.* The proof is similar to that of Theorem 3, except that since the myopic policy is with respect to $I'_G$ rather than $I_G$, on any episode, the maximum number of mispredictions $n$ is bounded above by $-\log(I'_G(g))$. Hence, the average number of mispredictions is given by:

$$\sum_g \ I_G(g)\log(\frac{1}{I'_G(g)}) = \sum_g I_G(g)\left(\log(\frac{1}{I'_G(g)}) + \log(I_G(g)) - \log(I_G(g))\right) =$$

$$\sum_g \ I_G(g)\log(\frac{I_G(g)}{I'_G(g)}) - \sum_g I_G(g)\log(I_G(g)) = \mathcal{H}(I_G) + \mathrm{KL}(I_G \parallel I'_G). \qquad \square$$

*Proof of Theorem 7.* According to the theory of POMDPs, the optimal action in a POMDP maximizes the sum of the immediate expected reward and the value of the resulting belief state (of the assistant) [6]. When the agent policy is deterministic, the initial goal distribution $I_G$ and the history of agent actions and states $H$ fully capture the belief state of the agent. Let $V(I_G, H)$ represent the optimal value of the current belief state. It satisfies the following Bellman equation, where $H'$ stands for the history after the assistant's action $h_i$ and the agent's action $a_j$.

$$V(I_G, H) = \max_{h_i} E(R((s, h_i), g, a_j)) + V(I_G, H')$$

Since there is only one agent's action $a^*(s, g)$ in $\Pi(s, g)$, the subsequent state $s'$ in $H'$, and its value do not depend on $h_i$. Hence the best helper action $h^*$ of the assistant is given by:

$$h^*(I_G, H) = \arg\max_{h_i} E(R((s, h_i), g, a^*(s, g))) = \arg\max_{h_i} \sum_{g \in C(H)} I_G(g)I(a_i \in \Pi(s, g))$$

$$= \arg\max_{h_i} I_G(C(H) \cap G(s, a_i))$$

where $C(H)$ is the set of goals consistent with the current history $H$, and $G(s, a_i)$ is the set of goals $g$ for which $a_i \in \Pi(s, g)$. $I(a_i \in \Pi(s, g))$ is an indicator function which is $= 1$ if $a_i \in \Pi(s, g)$. Note that $h^*$ is exactly the myopic policy. $\qquad \square$

*Proof of Theorem 9.* We first show that the problem is in NP. We build a tree representation of an optimal history-dependent policy for each initial state which acts as a polynomial-size certificate. Every node in the tree is represented by a pair $(s_i, G_i)$, where $s_i$ is a state and $G_i$ is a set of goals for which the node is on a good path from the root node. We let $h_i$ be the helper action selected in node $i$. The children of a node in the tree represent possible successor nodes $(s_j, G_j)$ reached by the agent's response to $h_i$. Note that multiple children can result from the same action because the dynamics is a function of the agent's goal.

To verify that the optimal policy tree is of polynomial size we note that the number of leaf nodes is upper bounded by $|G| \times \max_g N(g)$, where $N(g)$ is the number of leaf nodes generated by the goal $g$ and $G$ is the set of all goals. To estimate $N(g)$, we note that by our protocol, for any node $(s_i, G_i)$ where $g \in G_i$ and the assistant's action is $h_i$, if $a_i \in \Pi(s, g)$, it will have a single successor that contains $g$. Otherwise, there is a misprediction, which leads to at most $k$ successors for $g$. Hence, the number of nodes reached for $g$ grows geometrically with the number of mispredictions. Since there are at most $\log|G|$ mispredictions in any such path, $N(g) \le k^{\log_2 |G|} = k^{\log_k |G| \log_2 k} = |G|^{\log_2 k}$. Hence the total number of all leaf nodes of the tree is bounded by $|G|^{1+\log k}$, and the total number of nodes in the tree is bounded by $m|G|^{1+\log k}$, where $m$ is the number of steps to the horizon. Since this is polynomial in the problem parameters, the problem is in NP.

To show NP-hardness, we reduce 3-SAT to the given problem. We consider each 3-literal clause $C_i$ of a propositional formula $\Phi$ as a possible goal. The rest of the proof is identical to that of Theorem 1 except that all variables are set by the assistant. The agent accepts every setting, except possibly the last one which he reverses if the clause evaluates to 0. Since the assistant does not get any useful information until it makes the clause true or fails to do so, its optimal policy is to choose the assignment that maximizes the number of satisfied clauses so that the mistakes are minimized. The assistant makes a single prediction mistake on the last literal of each clause that is not satisfied by the assignment. Hence, the worst regret on any goal is 0 iff the 3-SAT problem is satisfiable. $\qquad \square$

## Acknowledgments

The authors gratefully acknowledge the support of NSF under grants IIS-0905678 and IIS-0964705.

## Footnotes

[1] If there are multiple initial states, we build an OTT for each initial state. Then the rank would be the maximum of the ranks of all trees.

# References

[1] Xinlong Bao, Jonathan L. Herlocker, and Thomas G. Dietterich. Fewer clicks and less frustration: reducing the cost of reaching the right folder. In *IUI*, pages 178–185, 2006.

[2] J. Boger, P. Poupart, J. Hoey, C. Boutilier, G. Fernie, and A. Mihailidis. A decision-theoretic approach to task assistance for persons with dementia. In *IJCAI*, 2005.

[3] Anton N. Dragunov, Thomas G. Dietterich, Kevin Johnsrude, Matt McLaughlin, Lida Li, and Jon L. Herlocker. Tasktracer: A desktop environment to support multi-tasking knowledge workers. In *Proceedings of IUI*, 2005.

[4] Andrzej Ehrenfeucht and David Haussler. Learning decision trees from random examples. *Information and Computation*, 82(3):231–246, September 1989.

[5] A. Fern, S. Natarajan, K. Judah, and P. Tadepalli. A decision-theoretic model of assistance. In *Proceedings of the International Joint Conference in AI*, 2007.

[6] Leslie Pack Kaelbling, Michael L. Littman, and Anthony R. Cassandra. Planning and acting in partially observable stochastic domains. *Artificial Intelligence*, 101:99–134, 1998.

[7] Tessa A. Lau, Steven A. Wolfman, Pedro Domingos, and Daniel S. Weld. Programming by demonstration using version space algebra. *Machine Learning*, 53(1-2):111–156, 2003.

[8] H. Lieberman. User interface goals, AI opportunities. *AI Magazine*, 30(2), 2009.

[9] M. L . Littman. *Algorithms for Sequential Decision Making*. PhD thesis, Brown University, Providence, RI, 1996.

[10] Martin Mundhenk. *The complexity of planning with partially-observable Markov Decision Processes*. PhD thesis, Friedrich-Schiller-Universitdt, 2001.

[11] K. Myers, P. Berry, J. Blythe, K. Conley, M. Gervasio, D. McGuinness, D. Morley, A. Pfeffer, M. Pollack, and M. Tambe. An intelligent personal assistant for task and time management. *AI Magazine*, 28(2):47–61, 2007.

[12] C. Papadimitriou and J. Tsitsiklis. The complexity of Markov Decision Processes. *Mathematics of Operations Research*, 12(3):441–450, 1987.

[13] M. Tambe. Electric Elves: What went wrong and why. *AI Magazine*, 29(2), 2008.

